# Generative and Discriminative Learning with Unknown Labeling Bias

**Miroslav Dudík**
Carnegie Mellon University
5000 Forbes Ave, Pittsburgh, PA 15213
mdudik@cmu.edu

**Steven J. Phillips**
AT&T Labs − Research
180 Park Ave, Florham Park, NJ 07932
phillips@research.att.com

## Abstract

We apply robust Bayesian decision theory to improve both generative and discriminative learners under bias in class proportions in labeled training data, when the true class proportions are unknown. For the generative case, we derive an entropy-based weighting that maximizes expected log likelihood under the worst-case true class proportions. For the discriminative case, we derive a multinomial logistic model that minimizes worst-case conditional log loss. We apply our theory to the modeling of species geographic distributions from presence data, an extreme case of labeling bias since there is no absence data. On a benchmark dataset, we find that entropy-based weighting offers an improvement over constant estimates of class proportions, consistently reducing log loss on unbiased test data.

## 1 Introduction

In many real-world classification problems, it is not equally easy or affordable to verify membership in different classes. Thus, class proportions in labeled data may significantly differ from true class proportions. In an extreme case, labeled data for an entire class might be missing (for example, negative experimental results are typically not published). A naively trained learner may perform poorly on test data that is not similarly afflicted by labeling bias. Several techniques address labeling bias in the context of cost-sensitive learning and learning from imbalanced data [5, 11, 2]. If the labeling bias is known or can be estimated, and all classes appear in the training set, a model trained on biased data can be corrected by reweighting [5]. When the labeling bias is unknown, a model is often selected using threshold-independent analysis such as ROC curves [11]. A good ROC curve, however, does not guarantee a low loss on test data. Here, we are concerned with situations when the labeling bias is unknown and some classes may be missing, but we have access to unlabeled data. We want to construct models that in addition to good ROC-based performance, also yield low test loss. We will be concerned with minimizing joint and conditional log loss, or equivalently, maximizing joint and conditional log likelihood.

Our work is motivated by the application of modeling species' geographic distributions from occurrence data. The data consists of a set of locations within some region (for example, the Australian wet tropics) where a species (such as the golden bowerbird) was observed, and a set of features such as precipitation and temperature, describing environmental conditions at each location. Species distribution modeling suffers from extreme imbalance in training data: we often only have information about species presence (positive examples), but no information about species absence (negative examples). We do, however, have unlabeled data, obtained either by randomly sampling locations from the region [4], or pooling presence data for several species collected with similar methods to yield a representative sample of locations which biologists have surveyed [13].

Previous statistical methods for species distribution modeling can be divided into three main approaches. The first interprets all unlabeled data as examples of species absence and learns a rule

to discriminate them from presences [19, 4]. The second embeds a discriminative learner in the EM algorithm in order to infer presences and absences in unlabeled data; this explicitly requires knowledge of true class probabilities [17]. The third models the presences alone, which is known in machine learning as one-class estimation [14, 7]. When using the first approach, the training data is commonly reweighted so that positive and negative examples have the same weight [4]; this models a quantity monotonically related to conditional probability of presence [13], with the relationship depending on true class probabilities. If we use $y$ to denote the binary variable indicating presence and $x$ to denote a location on the map, then the first two approaches yield models of conditional probability $p(y = 1|x)$, given estimates of true class probabilities. On the other hand, the main instantiation of the third approach, maximum entropy density estimation (maxent) [14] yields a model of the distribution $p(x|y = 1)$. To convert this to an estimate of $p(y = 1|x)$ (as is usually required, and necessary for measuring conditional log loss on which we focus here) again requires knowledge of the class probabilities $p(y = 1)$ and $p(y = 0)$. Thus, existing discriminative approaches (the first and second) as well as generative approaches (the third) require estimates of true class probabilities.

We apply robust Bayesian decision theory, which is closely related to the maximum entropy principle [6], to derive conditional probability estimates $p(y \mid x)$ that perform well under a wide range of test distributions. Our approach can be used to derive robust estimates of class probabilities $p(y)$ which are then used to reweight discriminative models or to convert generative models into discriminative ones. We present a treatment for the general multiclass problem, but our experiments focus on one-class estimation and species distribution modeling in particular. Using an extensive evaluation on real-world data, we show improvement in both generative and discriminative techniques.

Throughout this paper we assume that the difficulty of uncovering the true class label depends on the class label $y$ alone, but is independent of the example $x$. Even though this assumption is simplistic, we will see that our approach yields significant improvements. A related set of techniques estimates and corrects for the bias in sample selection, also known as covariate shift [9, 16, 18, 1, 13]. When the bias can be decomposed into an estimable and inestimable part, the right approach might be to use a combination of techniques presented in this paper and those for sample-selection bias.

## 2 Robust Bayesian Estimation with Unknown Class Probabilities

Our goal is to estimate an unknown conditional distribution $\pi(y \mid x)$, where $x \in \mathcal{X}$ is an example and $y \in \mathcal{Y}$ is a label. The input consists of labeled examples $(x_1, y_1), \ldots, (x_m, y_m)$ and unlabeled examples $x_{m+1}, \ldots, x_M$. Each example $x$ is described by a set of features $f_j : \mathcal{X} \to \mathbb{R}$, indexed by $j \in \mathcal{J}$. For simplicity, we assume that sets $\mathcal{X}$, $\mathcal{Y}$, and $\mathcal{J}$ are finite, but we would like to allow the space $\mathcal{X}$ and the set of features $\mathcal{J}$ to be very large.

In species distribution modeling from occurrence data, the space $\mathcal{X}$ corresponds to locations on the map, features are various functions derived from the environmental variables, and the set $\mathcal{Y}$ contains two classes: presence ($y = 1$) and absence ($y = 0$) for a particular species. Labeled examples are presences of the species, e.g., recorded presence locations of the golden bowerbird, while unlabeled examples are locations that have been surveyed by biologists, but neither presence nor absence was recorded. The unlabeled examples can be obtained as presence locations of species observed by a similar protocol, for example other birds [13].

We posit a joint density $\pi(x, y)$ and assume that examples are generated by the following process. First, a pair $(x, y)$ is chosen according to $\pi$. We always get to see the example $x$, but the label $y$ is revealed with an unknown probability that depends on $y$ and is independent of $x$. This means that we have access to independent samples from $\pi(x)$ and from $\pi(x \mid y)$, but no information about $\pi(y)$. In our example, species presence is revealed with an unknown fixed probability whereas absence is revealed with probability zero (i.e., never revealed).

### 2.1 Robust Bayesian Estimation, Maximum Entropy, and Logistic Regression

Robust Bayesian decision theory formulates an estimation problem as a zero-sum game between a decision maker and nature [6]. In our case, the decision maker chooses an estimate $p(x, y)$ while nature selects a joint density $\pi(x, y)$. Using the available data, the decision maker forms a set $\mathcal{P}$ in which he believes nature's choice lies, and tries to minimize worst-case loss under nature's choice. In this paper we are interested in minimizing the worst-case log loss relative to a fixed default

estimate $\nu$ (equivalently, maximizing the worst-case log likelihood ratio)

$$\min_{p \in \Delta} \max_{\pi \in \mathcal{P}} \mathbf{E}_\pi \left[ \ln \left( \frac{p(X, Y)}{\nu(X, Y)} \right) \right] \quad . \tag{1}$$

Here, $\Delta$ is the simplex of joint densities and $\mathbf{E}_\pi$ is a shorthand for $\mathbf{E}_{X,Y \sim \pi}$. The default density $\nu$ represents any prior information we have about $\pi$; if we have no prior information, $\nu$ is typically the uniform density.

Grünwald and Dawid [6] show that the robust Bayesian problem (Eq. 1) is often equivalent to the minimum relative entropy problem

$$\min_{p \in \mathcal{P}} \mathrm{RE}(p \,\|\, \nu) \quad , \tag{2}$$

where $\mathrm{RE}(p \,\|\, q) = \mathbf{E}_p[\ln(p(X, Y)/q(X, Y)]$ is relative entropy or Kullback-Leibler divergence and measures discrepancy between distributions $p$ and $q$. The formulation intuitively says that we should choose the density $p$ which is closest to $\nu$ while respecting constraints $\mathcal{P}$. When $\nu$ is uniform, minimizing relative entropy is equivalent to maximizing entropy $\mathrm{H}(p) = \mathbf{E}_p[-\ln p(X, Y)]$. Hence, the approach is mainly referred to as maximum entropy [10] or maxent for short. The next theorem outlines the equivalence of robust Bayes and maxent for the case considered in this paper. It is a special case of Theorem 6.4 of [6].

**Theorem 1** (Equivalence of maxent and robust Bayes). *Let $\mathcal{X} \times \mathcal{Y}$ be a finite sample space, $\nu$ a density on $\mathcal{X} \times \mathcal{Y}$ and $\mathcal{P} \subseteq \Delta$ a closed convex set containing at least one density absolutely continuous w.r.t. $\nu$. Then Eqs.* (1) *and* (2) *have the same optimizers.*

For the case without labeling bias, the set $\mathcal{P}$ is usually described in terms of equality constraints on moments of the joint distribution (feature expectations). Specifically, feature expectations with respect to $p$ are required to equal their empirical averages. When features are functions of $x$, but the goal is to discriminate among classes $y$, it is natural to consider a derived set of features which are versions of $f_j(x)$ active solely in individual classes $y$ (see for instance [8]). If we were to estimate the distribution of the golden bowerbird from presence-absence data then moment equality constraints require that the joint model $p(x, y)$ match the average altitude of presence locations as well as the average altitude of absence locations (both weighted by their respective training proportions).

When the number of samples is too small or the number of features too large then equality constraints lead to overfitting because the true distribution does not match empirical averages exactly. Overfitting is alleviated by relaxing the constraints so that feature expectations are only required to lie within a certain distance of sample averages [3].

The solution of Eq. (2) with equality or relaxed constraints can be shown to lie in an exponential family parameterized by $\boldsymbol{\lambda} = \langle \boldsymbol{\lambda}^y \rangle_{y \in \mathcal{Y}}$, $\boldsymbol{\lambda}^y \in \mathbb{R}^\partial$, and containing densities

$$q_{\boldsymbol{\lambda}}(x, y) \propto \nu(x, y) e^{\boldsymbol{\lambda}^y \cdot \boldsymbol{f}(x)} \quad .$$

The optimizer of Eq. (2) is the unique density which minimizes the empirical log loss

$$\frac{1}{m} \sum_{i \leq m} \ln q_{\boldsymbol{\lambda}}(x_i, y_i) \tag{3}$$

possibly with an additional $\ell_1$-regularization term accounting for slacks in equality constraints. (See [3] for a proof.)

In addition to constraints on moments of the joint distribution, it is possible to introduce constraints on marginals of $p$. The most common implementations of maxent impose marginal constraints $p(x) = \tilde{\pi}^{\mathrm{lab}}(x)$ where $\tilde{\pi}^{\mathrm{lab}}$ is the empirical distribution over labeled examples. The solution then takes form $q_{\boldsymbol{\lambda}}(x, y) = \tilde{\pi}^{\mathrm{lab}}(x) q_{\boldsymbol{\lambda}}(y \,|\, x)$ where $q_{\boldsymbol{\lambda}}(y \,|\, x)$ is the multinomial logistic model

$$q_{\boldsymbol{\lambda}}(y \,|\, x) \propto \nu(y \,|\, x) e^{\boldsymbol{\lambda}^y \cdot \boldsymbol{f}(x)} \quad .$$

As before, the maxent solution is the unique density of this form which minimizes the empirical log loss (Eq. 3). The minimization of Eq. (3) is equivalent to the minimization of conditional log loss

$$\frac{1}{m} \sum_{i \leq m} -\ln q_{\boldsymbol{\lambda}}(y_i \,|\, x_i) \quad .$$

Hence, this approach corresponds to logistic regression. Since it only models the labeling process $\pi(y \,|\, x)$, but not the sample generation $\pi(x)$, it is known as discriminative training.

The case with equality constraints $p(y) = \tilde{\pi}^{\mathrm{lab}}(y)$ has been analyzed for example by [8]. The solution has the form $q_{\boldsymbol{\lambda}}(x, y) = \tilde{\pi}^{\mathrm{lab}}(y) q_{\boldsymbol{\lambda}}(x \,|\, y)$ with

$$q_{\boldsymbol{\lambda}}(x \,|\, y) \propto \nu(x \,|\, y) e^{\boldsymbol{\lambda}^y \cdot \boldsymbol{f}(x)} \ .$$

Log loss can be minimized for each class separately, i.e., each $\boldsymbol{\lambda}^y$ is the maximum likelihood estimate (possibly with regularization) of $\pi(x \,|\, y)$. The joint estimate $q_{\boldsymbol{\lambda}}(x, y)$ can be used to derive the conditional distribution $q_{\boldsymbol{\lambda}}(y \,|\, x)$. Since this approach estimates the sample generating distributions of individual classes, it is known as generative training. Naive Bayes is a special case of generative training when $\nu(x \,|\, y) = \prod_j \nu_j(f_j(x) \,|\, y)$.

The two approaches presented in this paper can be viewed as generalizations of generative and discriminative training with two additional components: availability of unlabeled examples and lack of information about class probabilities. The former will influence the choice of the default $\nu$, the latter the form of constraints $\mathcal{P}$.

## 2.2 Generative Training: Entropy-weighted Maxent

When the number of labeled and unlabeled examples is sufficiently large, it is reasonable to assume that the empirical distribution $\tilde{\pi}(x)$ over all examples (labeled and unlabeled) is a faithful representation of $\pi(x)$. Thus, we consider defaults with $\nu(x) = \tilde{\pi}(x)$, shown to work well in species distribution modeling [13]. For simplicity, we assume that $\nu(y \,|\, x)$ does not depend on $x$ and focus on $\nu(x, y) = \tilde{\pi}(x)\nu(y)$. Other options are possible. For example, when the number of examples is small, $\tilde{\pi}(x)$ might be replaced by an estimate of $\pi(x)$. The distribution $\nu(y)$ can be chosen uniform across $y$, but if some classes are known to be rarer than others then a non-uniform estimate will perform better. In Section 3, we analyze the impact of this choice.

Constraints on moments of the joint distribution, such as those in the previous section, will misspecify true moments in the presence of labeling bias. However, as discussed earlier, labeled examples from each class $y$ approximate conditional distributions $\pi(x \,|\, y)$. Thus, instead of constraining joint expectations, we constrain conditional expectations $\mathbf{E}_p[f_j(X) \,|\, y]$. In general, we consider robust Bayes and maxent problems with the set $\mathcal{P}$ of the form $\mathcal{P} = \{p \in \Delta : p_{\mathcal{X}}^y \in \mathcal{P}_{\mathcal{X}}^y\}$ where $p_{\mathcal{X}}^y$ denotes the $|\mathcal{X}|$-dimensional vector of conditional probabilities $p(x \,|\, y)$ and $\mathcal{P}_{\mathcal{X}}^y$ expresses the constraints on $p_{\mathcal{X}}^y$. For example, relaxed constraints for class $y$ are expressed as

$$\forall j : \ \left| \mathbf{E}_p[f_j(X) \,|\, y] - \tilde{\mu}_j^y \right| \leq \beta_j^y \tag{4}$$

where $\tilde{\mu}_j^y$ is the empirical average of $f_j$ among labeled examples in class $y$ and $\beta_j^y$ are estimates of deviations of averages from true expectations. Similar to [14], we use standard-error-like deviation estimates $\beta_j^y = \beta \tilde{\sigma}_j^y / \sqrt{m_y}$ where $\beta$ is a single tuning constant, $\tilde{\sigma}_j^y$ is the empirical standard deviation of $f_j$ among labeled examples in class $y$, and $m_y$ is the number of labeled examples in class $y$. When $m_y$ equals 0, we choose $\beta_j^y = \infty$ and thus leave feature expectations unconstrained.

The next theorem and the following corollary show that robust Bayes (and also maxent) with the constraint set $\mathcal{P}$ of the form above yield estimators similar to generative training. In addition to the notation $p_{\mathcal{X}}^y$ for conditional densities, we use the notation $p_{\mathcal{Y}}$ and $p_{\mathcal{X}}$ to denote vectors of marginal probabilities $p(y)$ and $p(x)$, respectively. For example, the empirical distribution over examples is denoted $\tilde{\pi}_{\mathcal{X}}$.

**Theorem 2.** *Let $\mathcal{P}_{\mathcal{X}}^y$, $y \in \mathcal{Y}$ be closed convex sets of densities over $\mathcal{X}$ and $\mathcal{P} = \{p \in \Delta : p_{\mathcal{X}}^y \in \mathcal{P}_{\mathcal{X}}^y\}$. If $\mathcal{P}$ contains at least one density absolutely continuous w.r.t. $\nu$ then robust Bayes and maxent over $\mathcal{P}$ are equivalent. The solution $\hat{p}$ has the form $\hat{p}(y)\hat{p}(x \,|\, y)$ where class-conditional densities $\hat{p}_{\mathcal{X}}^y$ minimize $\mathrm{RE}(p_{\mathcal{X}}^y \,\|\, \tilde{\pi}_{\mathcal{X}})$ among $p_{\mathcal{X}}^y \in \mathcal{P}_{\mathcal{X}}^y$ and*

$$\hat{p}(y) \propto \nu(y) e^{-\mathrm{RE}(\hat{p}_{\mathcal{X}}^y \,\|\, \tilde{\pi}_{\mathcal{X}})} \ . \tag{5}$$

*Proof.* It is not too difficult to verify that the set $\mathcal{P}$ is a closed convex set of joint densities, so the equivalence of robust Bayes and maxent follows from Theorem 1. To prove the remainder, we rewrite the maxent objective as

$$\mathrm{RE}(p \,\|\, \nu) = \mathrm{RE}(p_{\mathcal{Y}} \,\|\, \nu_{\mathcal{Y}}) + \sum_y p(y) \mathrm{RE}(p_{\mathcal{X}}^y \,\|\, \tilde{\pi}_{\mathcal{X}}) \ .$$

Maxent problem is then equivalent to

$$\min_{p_{\mathcal{Y}}}\Big[\mathrm{RE}(p_{\mathcal{Y}}\,\|\,\nu_{\mathcal{Y}}) + \sum_y p(y)\min_{p_{\mathcal{X}}^y\in\mathcal{P}_{\mathcal{X}}^y}\mathrm{RE}(p_{\mathcal{X}}^y\,\|\,\tilde{\pi}_{\mathcal{X}})\Big]$$

$$= \min_{p_{\mathcal{Y}}}\left[\left(\sum_y p(y)\ln\left(\frac{p(y)}{\nu(y)}\right)\right) + \left(\sum_y p(y)\mathrm{RE}(\hat{p}_{\mathcal{X}}^y\,\|\,\tilde{\pi}_{\mathcal{X}})\right)\right]$$

$$= \min_{p_{\mathcal{Y}}}\left[\sum_y p(y)\ln\left(\frac{p(y)}{\nu(y)e^{-\mathrm{RE}(\hat{p}_{\mathcal{X}}^y\,\|\,\tilde{\pi}_{\mathcal{X}})}}\right)\right]$$

$$= \mathrm{const.} + \min_{p_{\mathcal{Y}}}\mathrm{RE}(p_{\mathcal{Y}}\,\|\,\hat{p}_{\mathcal{Y}})\ .$$

Since $\mathrm{RE}(p\,\|\,q)$ is minimized for $p = q$, we indeed obtain that for the minimizing $p$, $p_{\mathcal{Y}} = \hat{p}_{\mathcal{Y}}$.　□

Theorem 2 generalizes to the case when in addition to constraining $p_{\mathcal{X}}^y$ to lie in $\mathcal{P}_{\mathcal{X}}^y$, we also constrain $p_{\mathcal{Y}}$ to lie in a closed convex set $\mathcal{P}_{\mathcal{Y}}$. The solution then takes form $p(y)\hat{p}(x\,|\,y)$ with $\hat{p}(x\,|\,y)$ as in the theorem, but with $p(y)$ minimizing $\mathrm{RE}(p_{\mathcal{Y}}\,\|\,\hat{p}_{\mathcal{Y}})$ subject to $p_{\mathcal{Y}} \in \mathcal{P}_{\mathcal{Y}}$. Unlike generative training without labeling bias, the class-conditional densities in the theorem above influence class probabilities. When sets $\mathcal{P}_{\mathcal{X}}^y$ are specified using constraints of Eq. (4) then $\hat{p}$ has a form derived from regularized maximum likelihood estimates in an exponential family (see, e.g., [3]):

**Corollary 3.** *If sets $\mathcal{P}_{\mathcal{X}}^y$ are specified by inequality constraints of Eq.* (4) *then robust Bayes and maxent are equivalent. The class-conditional densities $\hat{p}(x\,|\,y)$ of the solution take form*

$$q_{\boldsymbol{\lambda}}(x\,|\,y) \propto \tilde{\pi}(x)e^{\hat{\boldsymbol{\lambda}}^y\cdot\boldsymbol{f}(x)} \tag{6}$$

*and solve single-class regularized maximum likelihood problems*

$$\min_{\boldsymbol{\lambda}^y}\Big\{\sum_{i:y_i=y}\big[-\ln q_{\boldsymbol{\lambda}}(x_i\,|\,y)\big] + m_y\sum_{j\in\mathcal{J}}\beta_j|\lambda_j^y|\Big\}\ . \tag{7}$$

**One-class Estimation.** In one-class estimation problems, there are two classes (0 and 1), but we only have access to labeled examples from one class (e.g., class 1). In species distribution modeling, we only have access to presence records of the species. Based on labeled examples, we derive a set of constraints on $p(x\,|\,y = 1)$, but leave $p(x\,|\,y = 0)$ unconstrained. By Theorem 2, $\hat{p}(x\,|\,y = 1)$ then solves the single-class maximum entropy problem, we write $\hat{p}(x\,|\,y = 1) = \hat{p}_{\mathrm{ME}}(x)$, and $\hat{p}(x\,|\,y = 0) = \tilde{\pi}(x)$. Assume without loss of generality that examples $x_1,\dots,x_M$ are distinct (but allow them to have identical feature vectors). Thus, $\tilde{\pi}(x) = 1/M$ on examples and zero elsewhere, and $\mathrm{RE}(\hat{p}_{\mathrm{ME}}\,\|\,\tilde{\pi}_{\mathcal{X}}) = -\mathrm{H}(\hat{p}_{\mathrm{ME}}) + \ln M$. Plugging these into Theorem 2, we can derive the conditional estimate $\hat{p}(y = 1\,|\,x)$ across all unlabeled examples $x$:

$$\hat{p}(y = 1\,|\,x) = \frac{\nu(y = 1)\hat{p}_{\mathrm{ME}}(x)e^{\mathrm{H}(\hat{p}_{\mathrm{ME}})}}{\nu(y = 0) + \nu(y = 1)\hat{p}_{\mathrm{ME}}(x)e^{\mathrm{H}(\hat{p}_{\mathrm{ME}})}}\ . \tag{8}$$

If constraints on $p(x\,|\,y = 1)$ are chosen as in Corollary 3 then $\hat{p}_{\mathrm{ME}}$ is exponential and Eq. (8) thus describes a logistic model. This model has the same coefficients as $\hat{p}_{\mathrm{ME}}$, with the intercept chosen so that "typical" examples $x$ under $\hat{p}_{\mathrm{ME}}$ (examples with log probability close to the expected log probability) yield predictions close to the default.

### 2.3 Discriminative Training: Class-robust Logistic Regression

Similar to the previous section, we consider $\nu(x, y) = \tilde{\pi}(x)\nu(y)$. The set of constraints $\mathcal{P}$ will now also include equality constraints on $p(x)$. Since $\tilde{\pi}^{\mathrm{lab}}(x)$ misspecifies the marginal, we use $p(x) = \tilde{\pi}(x)$. Next theorem is an analog of Corollary 3 for discriminative training. It follows from a combination of Theorem 1 and duality of maxent with maximum likelihood [3]. A complete proof will appear in the extended version of this paper.

**Theorem 4.** *Assume that sets $\mathcal{P}_{\mathcal{X}}^y$ are specified by inequality constraints of Eq.* (4). *Let $\mathcal{P} = \{p \in \Delta : p_{\mathcal{X}}^y \in \mathcal{P}_{\mathcal{X}}^y \text{ and } p_{\mathcal{X}} = \tilde{\pi}_{\mathcal{X}}\}$. If the set $\mathcal{P}$ is non-empty then robust Bayes and maxent over $\mathcal{P}$ are equivalent. For the solution $\hat{p}$, $\hat{p}(x) = \tilde{\pi}(x)$ and $\hat{p}(y\,|\,x)$ takes form*

$$q_{\boldsymbol{\lambda}}(y\,|\,x) \propto \nu(y)e^{\boldsymbol{\lambda}^y\cdot\boldsymbol{f}(x) - \boldsymbol{\lambda}^y\cdot\bar{\boldsymbol{\mu}}^y + \sum_j \beta_j^y|\lambda_j^y|} \tag{9}$$

*and solves the regularized "logistic regression" problem*

$$\min_{\boldsymbol{\lambda}} \left\{ \frac{1}{M} \sum_{i \leq M} \sum_{y \in \mathcal{Y}} \left[ -\bar{\pi}(y \mid x_i) \ln q_{\boldsymbol{\lambda}}(y \mid x_i) \right] + \sum_{y \in \mathcal{Y}} \bar{\pi}(y) \sum_{j \in \mathcal{J}} \left[ \beta_j^y |\lambda_j^y| + (\bar{\mu}_j^y - \tilde{\mu}_j^y)\lambda_j^y \right] \right\} \quad . \quad (10)$$

*where $\bar{\pi}$ is an arbitrary feasible point, $\bar{\pi} \in \mathcal{P}$, and $\bar{\mu}_j^y$ its class-conditional feature expectations.*

We put logistic regression in quotes, because the model described by Eq. (9) is not the usual logistic model; however, once the parameters $\boldsymbol{\lambda}^y$ are fixed, Eq. (9) simply determines a logistic model with a special form of the intercept. Note that the second term of Eq. (10) is indeed a regularization, albeit possibly an asymmetric one, since any feasible $\bar{\pi}$ will have $|\bar{\mu}_j^y - \tilde{\mu}_j^y| \leq \beta_j^y$. Since $\bar{\pi}(x) = \tilde{\pi}(x)$, $\bar{\pi}$ is specified solely by $\bar{\pi}(y \mid x)$ and thus can be viewed as a tentative imputation of labels across all examples. We remark that the value of the objective of Eq. (10) does not depend on the choice of $\bar{\pi}$, because a different choice of $\bar{\pi}$ (influencing the first term) yields a different set of means $\bar{\mu}_j^y$ (influencing the second term) and these differences cancel out. To provide a more concrete example and some intuition about Eq. (10), we now consider one-class estimation.

**One-class estimation.** A natural choice of $\bar{\pi}$ is the "pseudo-empirical" distribution which views all unlabeled examples as negatives. Pseudo-empirical means of class 1 match empirical averages of class 1 exactly, whereas pseudo-empirical means of class 0 can be arbitrary because they are unconstrained. The lack of constraints on class 0 forces the corresponding $\boldsymbol{\lambda}^y$ to equal zero. The objective can thus be formulated solely using $\boldsymbol{\lambda}^y$ for the class 1; therefore, we will omit the superscript $y$. Eq. (10) after multiplying by $M$ then becomes

$$\min_{\boldsymbol{\lambda}} \left\{ \sum_{i \leq m} \left[ -\ln q_{\boldsymbol{\lambda}}(y = 1 \mid x_i) \right] + \sum_{m < i \leq M} \left[ -\ln q_{\boldsymbol{\lambda}}(y = 0 \mid x_i) \right] + m \sum_{j \in \mathcal{J}} \beta_j |\lambda_j| \right\} \quad .$$

Thus the objective of class-robust logistic regression is the same as of regularized logistic regression discriminating positives from unlabeled examples.

## 3    Experiments

We evaluate our techniques using a large real-world dataset containing 226 species from 6 regions of the world, produced by the "Testing alternative methodologies for modeling species' ecological niches and predicting geographic distributions" Working Group at the National Center for Ecological Analysis and Synthesis (NCEAS). The training set contains presence-only data from unplanned surveys or incidental records, including those from museums and herbariums. The test set contains presence-absence data from rigorously planned independent surveys (i.e., without labeling bias). The regions are described by 11–13 environmental variables, with 20–54 species per region, 2–5822 training presences per species (median of 57), and 102–19120 test points (presences and absences); for details see [4]. As unlabeled examples we use presences of species captured by similar methods, known as "target group", with the groups as in [13].

We evaluate both entropy-weighted maxent and class-robust logistic regression while varying the default estimate $\nu(y = 1)$, referred to as *default species prevalence* by analogy with $p(y = 1)$, which is called *species prevalence*. Entropy-weighted maxent solutions for different default prevalences are derived by Eq. (8) from the same one-class estimate $\hat{p}_{\mathrm{ME}}$. Class-robust logistic regression requires separate optimization for each default prevalence.

We calculate $\hat{p}_{\mathrm{ME}}$ using the *Maxent* package [15] with features spanning the space of piecewise linear splines (of each environmental variable separately) and a tuned value of $\beta$ (see [12] for the details on features and tuning). Class-robust logistic models are calculated by a boosting-like algorithm SUMMET [3] with the same set of features and the same value $\beta$ as the maxent runs.

For comparison, we also evaluate default-weighted maxent, using class probabilities $p(y) = \nu(y)$ instead of Eq. (5), and two "oracle" methods based on class probabilities in the test data: constant Bernoulli prediction $p(y \mid x) = \pi(y)$, and oracle-weighted maxent, using $p(y) = \pi(y)$ instead of Eq. (5). Note that the constant Bernoulli prediction has no discrimination power (its AUC is 0.5) even though it matches class probabilities perfectly.

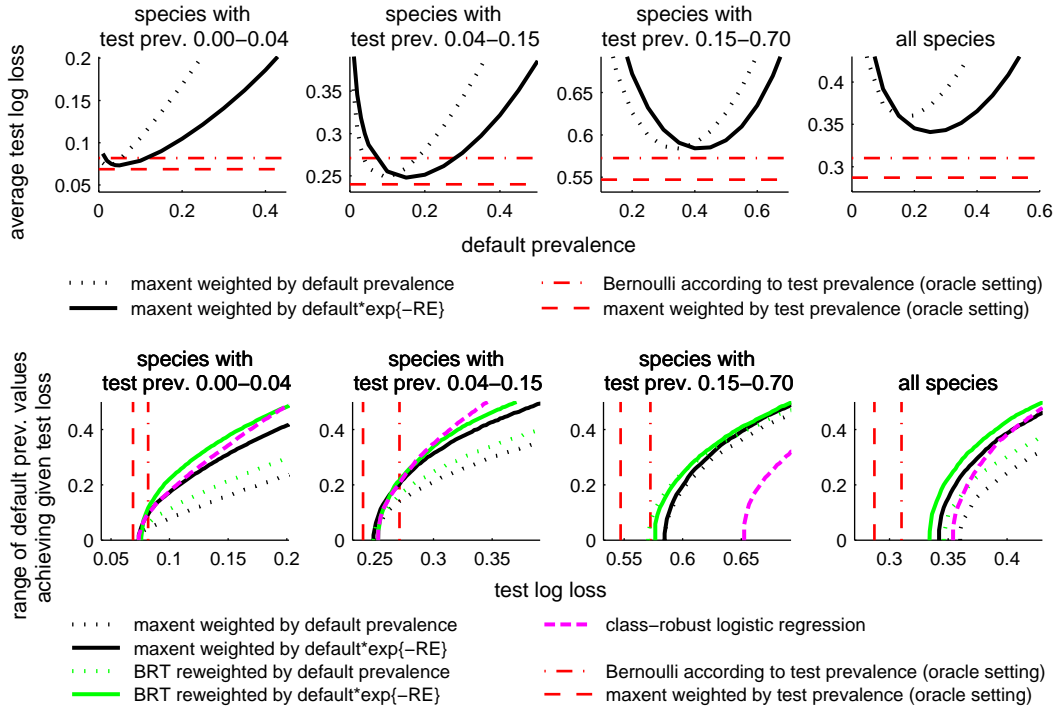

Figure 1: Comparison of reweighting schemes. *Top:* Test log loss averaged over species with given values of test prevalence, for varying default prevalence. *Bottom:* For each value of test log loss, we determine the range of default prevalence values that achieve it.

To test entropy-weighting as a general method for estimating class probabilities, we also evaluate boosted regression trees (BRT), which have the highest predictive accuracy along with maxent among species distribution modeling techniques [4]. In this application, BRT is used to construct a logistic model discriminating positive examples from unlabeled examples. Recent work [17] uses a more principled approach where unknown labels are fitted by an EM algorithm, but our preliminary runs had too low AUC values, so they are excluded from our comparison. We train BRT using the R package *gbm* on datasets weighted so that the total weight of positives is equal to the total weight of unlabeled examples, and then apply Elkan's reweighting scheme [5]. Specifically, the BRT result $\hat{p}_{\mathrm{BRT}}(y \mid x)$ is transformed to

$$p(y = 1 \mid x) = \frac{p(y = 1)\hat{p}_{\mathrm{BRT}}(y = 1 \mid x)}{p(y = 1)\hat{p}_{\mathrm{BRT}}(y = 1 \mid x) + p(y = 0)\hat{p}_{\mathrm{BRT}}(y = 0 \mid x)}$$

for two choices of $p(y)$: default, $p(y) = \nu(y)$, and entropy-based (using $\hat{p}_{\mathrm{ME}}$).

All three techniques yield state-of-the-art discrimination (see [13]) measured by the average AUC: maxent achieves AUC of 0.7583; class-robust logistic regression 0.7451–0.7568; BRT 0.7545. Unlike maxent and BRT estimates, class-robust logistic estimates are not monotonically related, so they yield different AUC for different default prevalence. However, log loss performance varies broadly according to the reweighting scheme. In the top portion of Fig. 1, we focus on maxent. Naive weighting by default prevalence yields sharp peaks in performance around the best default prevalence. Entropy-based weighting yields broader peaks, so it is less sensitive to the default prevalence. The improvement diminishes as the true prevalence increases, but entropy-based weighting is never more sensitive. Thanks to smaller sensitivity, entropy-based weighting outperforms naive weighting when a single default needs to be chosen for all species (the rightmost plot). Note that the optimal default values are higher for entropy-based weighting, because in one-class estimation the entropy-based prevalence is always smaller than default (unless the estimate $\hat{p}_{\mathrm{ME}}$ is uniform).

Improved sensitivity is demonstrated more clearly in the bottom portion of Fig. 1, now also including BRT and class-robust logistic regression. We see that BRT and maxent results are fairly similar, with BRT performing overall slightly better than maxent. Note that entropy-reweighted BRT relies both on BRT and maxent for its performance. A striking observation is the poor performance of class-robust logistic regression for species with larger prevalence values; it merits further investigation.

# 4  Conclusion and Discussion

To correct for unknown labeling bias in training data, we used robust Bayesian decision theory and developed generative and discriminative approaches that optimize log loss under worst-case true class proportions. We found that our approaches improve test performance on a benchmark dataset for species distribution modeling, a one-class application with extreme labeling bias.

**Acknowledgments.**  We would like to thank all of those who provided data used here: A. Ford, CSIRO Atherton, Australia; M. Peck and G. Peck, Royal Ontario Museum; M. Cadman, Bird Studies Canada, Canadian Wildlife Service of Environment Canada; the National Vegetation Survey Databank and the Allan Herbarium, New Zealand; Missouri Botanical Garden, especially R. Magill and T. Consiglio; and T. Wohlgemuth and U. Braendi, WSL Switzerland.

# References

[1] Bickel, S., M. Brückner, and T. Scheffer (2007).  Discriminative learning for differing training and test distributions. In *Proc. 24th Int. Conf. Machine Learning*, pp. 161–168.

[2] Chawla, N. V., N. Japkowicz, and A. Kołcz (2004).  Editorial: special issue on learning from imbalanced data sets. *SIGKDD Explorations 6*(1), 1–6.

[3] Dudík, M., S. J. Phillips, and R. E. Schapire (2007). Maximum entropy density estimation with generalized regularization and an application to species distribution modeling. *J. Machine Learning Res. 8*, 1217–1260.

[4] Elith, J., C. H. Graham, et al. (2006).  Novel methods improve prediction of species' distributions from occurrence data. *Ecography 29*(2), 129–151.

[5] Elkan, C. (2001).  The foundations of cost-sensitive learning. In *Proc. 17th Int. Joint Conf. on Artificial Intelligence*, pp. 973–978.

[6] Grünwald, P. D. and A. P. Dawid (2004).  Game theory, maximum entropy, minimum discrepancy, and robust Bayesian decision theory. *Ann. Stat. 32*(4), 1367–1433.

[7] Guo, Q., M. Kelly, and C. H. Graham (2005). Support vector machines for predicting distribution of Sudden Oak Death in California. *Ecol. Model. 182*, 75–90.

[8] Haffner, P., S. Phillips, and R. Schapire (2005).  Efficient multiclass implementations of L1-regularized maximum entropy. E-print arXiv:cs/0506101.

[9] Heckman, J. J. (1979). Sample selection bias as a specification error. *Econometrica 47*(1), 153–161.

[10] Jaynes, E. T. (1957). Information theory and statistical mechanics. *Phys. Rev. 106*(4), 620–630.

[11] Maloof, M. (2003).  Learning when data sets are imbalanced and costs are unequal and unknown. In *Proc. ICML'03 Workshop on Learning from Imbalanced Data Sets*.

[12] Phillips, S. J. and M. Dudík (2008).  Modeling of species distributions with Maxent: new extensions and a comprehensive evaluation. *Ecography 31*(2), 161–175.

[13] Phillips, S. J., M. Dudík, J. Elith, C. H. Graham, A. Lehmann, J. Leathwick, and S. Ferrier.  Sample selection bias and presence-only models of species distributions: Implications for selection of background and pseudo-absences. *Ecol. Appl.* To appear.

[14] Phillips, S. J., M. Dudík, and R. E. Schapire (2004).  A maximum entropy approach to species distribution modeling. In *Proc. 21st Int. Conf. Machine Learning*, pp. 655–662. ACM Press.

[15] Phillips, S. J., M. Dudík, and R. E. Schapire (2007).  Maxent software for species habitat modeling. http://www.cs.princeton.edu/~schapire/maxent.

[16] Shimodaira, H. (2000). Improving predictive inference under covariate shift by weighting the log-likelihood function. *J. Stat. Plan. Infer. 90*(2), 227–244.

[17] Ward, G., T. Hastie, S. Barry, J. Elith, and J. Leathwick (2008).  Presence-only data and the EM algorithm. *Biometrics*. In press.

[18] Zadrozny, B. (2004).  Learning and evaluating classifiers under sample selection bias.  In *Proc. 21st Int. Conf. Machine Learning*, pp. 903–910. ACM Press.

[19] Zaniewski, A. E., A. Lehmann, and J. M. Overton (2002).  Predicting species spatial distributions using presence-only data: A case study of native New Zealand ferns. *Ecol. Model. 157*, 261–280.